# Example Based Image Synthesis of Articulated Figures

**Trevor Darrell**
Interval Research, 1801C Page Mill Road, Palo Alto CA 94304
trevor@interval.com, http://www.interval.com/~trevor/

## Abstract

We present a method for learning complex appearance mappings, such as occur with images of articulated objects. Traditional interpolation networks fail on this case since appearance is not necessarily a smooth function nor a linear manifold for articulated objects. We define an appearance mapping from examples by constructing a set of independently smooth interpolation networks; these networks can cover overlapping regions of parameter space. A set growing procedure is used to find example clusters which are well-approximated within their convex hull; interpolation then proceeds only within these sets of examples. With this method physically valid images are produced even in regions of parameter space where nearby examples have different appearances. We show results generating both simulated and real arm images.

## 1 Introduction

Image-based view synthesis is an important application of learning networks, offering the ability to render realistic images without requiring detailed models of object shape and illumination effects. To date, much attention has been given to the problem of view synthesis under varying camera pose or rigid object transformation. Several successful solutions have been proposed in the computer graphics and vision literature, including view morphing [12], plenoptic modeling/depth recovery [8], "lightfields" [7], and recent approaches using the trifocal tensor for view extrapolation [13].

For non-rigid view synthesis, networks for model-based interpolation and manifold learning have been used successfully in some cases [14, 2, 4, 11]. Techniques based on Radial Basis Function (RBF) interpolation or on Principle Components Analysis (PCA), have been able to interpolate face images under varying pose, expression and identity [1, 5, 6]. How-

extends the notion of example clustering to the case of coupled shape and texture appearance models.

Our basic method is to find sets of examples which can be well-approximated from their convex hull in parameter space. We define a set growing criterion which enforces compactness and the good-interpolation property. To add a new point to an example set, we require both that the new point must be well approximated by the previous set alone and that all interior points in the resulting set be well interpolated from the exterior examples. We define exterior examples to be those on the convex hull of the set in parameter space. Given a training subset $s \subset \Omega$ and new point $p \in \Omega$,

$$E(s, p) = \max(E_I(s \cup \{p\}), E_E(s, p)),$$

with the interior and extrapolation error defined as

$$E_I(s) = \max_{p' \in (s - \mathcal{H}_x(s))} \|y_{p'} - \hat{y}(\mathcal{H}_x(s), x_{p'})\|, \qquad E_E(s, p) = \|y_p - \hat{y}(\mathcal{H}_x(s), x_p)\|.$$

$\mathcal{H}_x(s)$ is the subset of $s$ whose $x$ vectors lie on the convex hull of all such vectors in $s$. To add a new point, we require $E < \epsilon$, where $\epsilon$ is a free parameter of the clustering method.

Given a seed example set, we look to nearest neighbors in appearance space to find the next candidate to add. Unless we are willing to test the extrapolation error of the current model to all points, we have to rely on precomputed non-vectorized appearance distance (e.g., MSE between example images). If the examples are sparse in the appearance domain, this may not lead to effective groupings.

If examples are provided in sequence and are based on observations from an object with realistic dynamics, then we can find effective groupings even if observations are sparse in appearance space. We make the assumption that along the trajectory of example observations over time, the underlying object is likely to remain smooth and locally span regions of appearance which are possible to interpolate. We thus perform set growing along examples on their input trajectory. Specifically, in the results reported below, we select $K$ seed points on the trajectory to form initial clusters. At each point $p$ we find the set $s$ which is the smallest interval on the example trajectory which contains $p$, has a non-zero interior region $(s - \mathcal{H}_x(s))$, and for which $E_I(s) < \epsilon$. If such set exists, we continue to expand it, growing the set along the example trajectory until the above set growing criterion is violated. Once we can no longer grow any set, we test whether any set is a proper subset of another, and delete it if so. We keep the remaining sets, and use them for interpolation as described below.

## 4 Synthesis using example sets

We generate new views using sets of examples: interpolation is restricted to only occur inside the convex hull of an example set found as above for which $E_I(s) \leq \epsilon$. Given a new parameter vector $x$, we test whether it is in the convex hull of parameters in any example set. If the point does not lie in the convex hull of any example set, we find the nearest point on the convex hull of one of the example sets, and use that instead. This prevents erroneous extrapolation.

If a new parameter is in the convex hull of more than one example set, we first select the set whose median example parameter is closest to the desired example parameter. Once a set has been selected, we interpolate a new function value from examples using the RBF method summarized above. To enforce temporal consistency of rendered images over time,

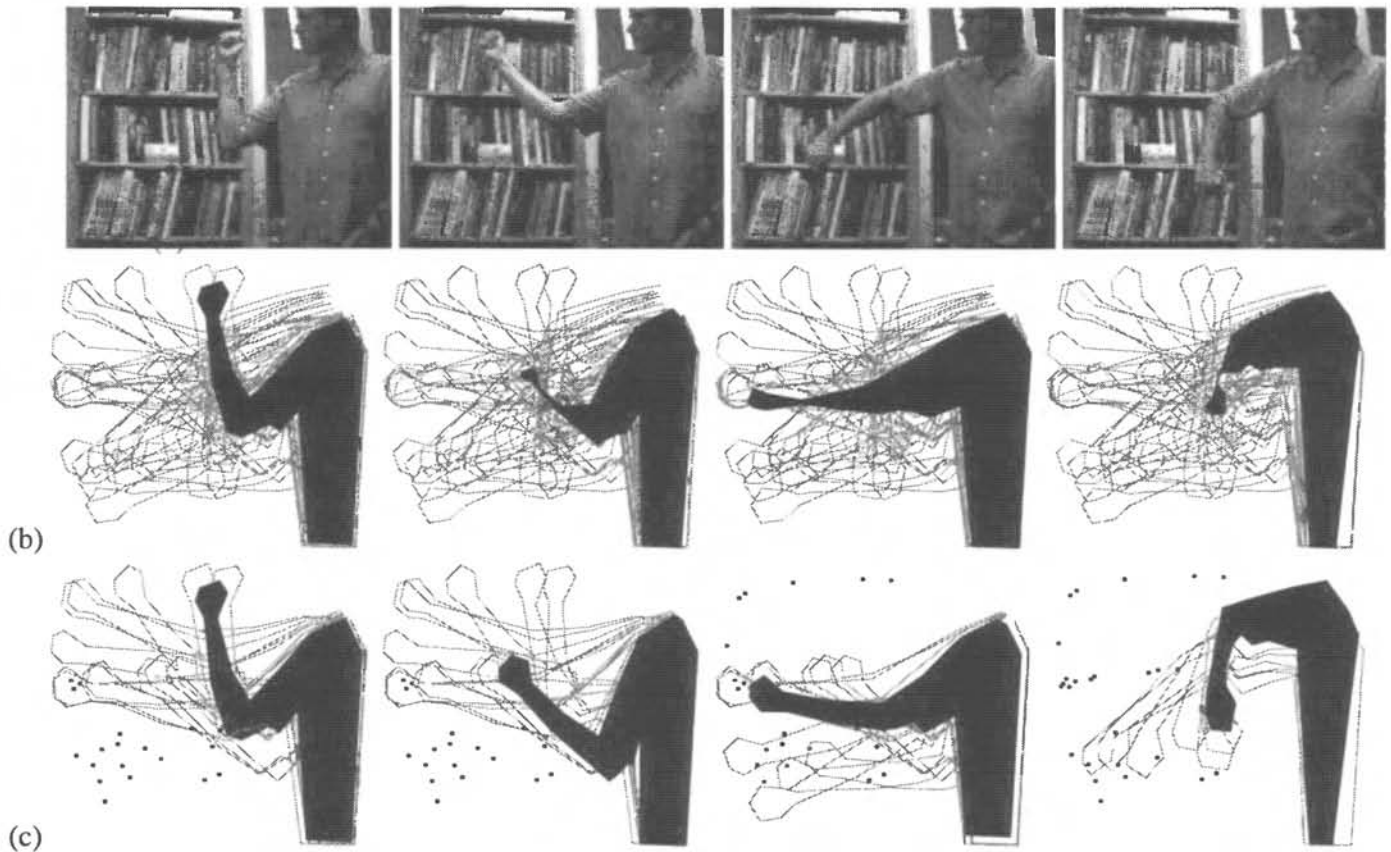

(b)

(c)

Figure 2: (a) Images of a real arm (from a sequence of 33 images) with changing appearance and elbow configuration. (b,c) Interpolated shape of arms tracked in previous figure. (b) shows results using all examples in a single interpolation network; (c) shows results using example sets algorithm. Open contours show arms example locations; filled contour shows interpolation result. Near regions of appearance singularity in parameter space the full network method generates physically-invalid arm shapes; the example sets method produces realistic images.

The method presented below for grouping examples into locally valid spaces is generally applicable to both the PCA and RBF-based view synthesis techniques. However our initial implementation, and the results reported in this paper, have been with RBF-based models.

## 3   Finding consistent example sets

Given examples from a complicated (non-linear, non-smooth) appearance mapping, we find local regions of appearance which are well-behaved as smooth, possibly linear, functions. We wish to cluster our examples into sets which can be used for successful interpolation using our local appearance model.

Conceptually, this problem is similar to that faced by Bregler and Omohundro [2], who built image manifolds using a mixture of local PCA models. Their work was limited to modeling shape (lip outlines); they used K-means clustering of image appearance to form the initial groupings for PCA analysis. However this approach had no model of texture and performed clustering using a mean-squared-error distance metric in simple appearance. Simple appearance clustering drastically over-partitions the appearance space compared to a model that jointly represent shape and texture. Examples which are distant in simple appearance can often be close when considered in 'vectorized' representation. Our work

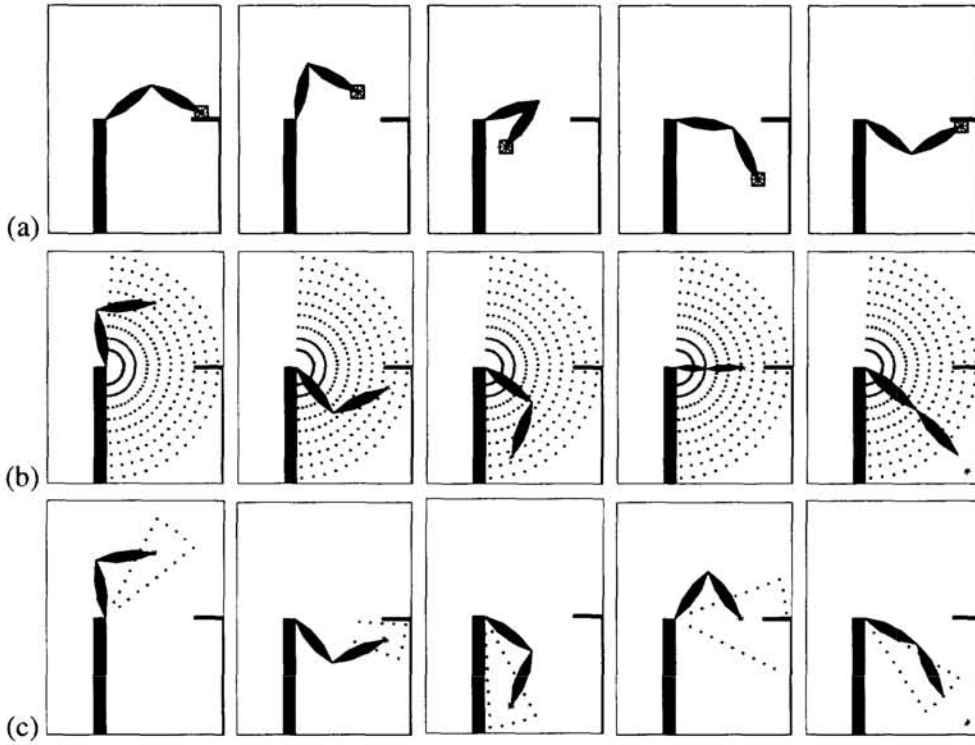

Figure 1: Arm appearance interpolated from examples using approximation network. (a) A 2DOF planar arm. Discontinuities in appearance due to workspace constraints make this a difficult function to learn from examples; the first and last example are very close in parameter space, but far in appearance space. (b) shows results using all examples in a single network; (c) using the example sets algorithm described in text. Note poor approximation on last two examples in (a); appearance discontinuities and extrapolation cause problems for full network, but are handled well in examples sets method.

In PCA-based approaches, $G$ projects a portion of $u$ onto a optimal linear subspace found from $D$, and $F$ projects a portion of $u$ onto a subspace found from $T$ [6, 5]. For example $G_D(u) = P_D^m S_g u$, where $S_g$ is a diagonal boolean matrix which selects the texture parameters from $u$ and $P_D^m$ is a matrix containing the $m$-th largest principle components of $D$. $F$ warps the reconstructed texture according to the given shape: $F_T(u, s) = [P_T^m S_t u] \circ s$. While interpolation is simple using a PCA approach, the parameters used in PCA models often do not have any direct physical interpretation. For the task of view synthesis, an additional mapping $u = H(x)$ is needed to map from task parameters to PCA input values; a backpropogation neural net was used to perform this function for the task of eye gaze analysis [10].

Using the RBF-based approach [1], the application to view synthesis is straightforward. Both $G$ and $F$ are networks which compute locally-weighted regression, and parameters are used directly ($u = x$). $G$ computes an interpolated shape, and $F$ warps and blends the example texture images according to that shape: $G_D(x) = \sum_i c_i f(x - x_i)$, $F_T(x, s) = [\sum_i c_i' f(x - x_i)] \circ s$, where $f$ is a radial basis function. The coefficients $c$ and $c'$ are derived from $D$ and $T$, respectively: $C = DR^+$, where $r_{ij} = f(x_i - x_j)$ and $C$ is the matrix of row vectors $c_i$; similarly $C' = TR^+$ [9]. We have found both vector norm and Gaussian basis functions give good results when appearance data is from a smooth function; the results below use $f(r) = \|r\|$.

ever, these methods are limited in the types of object appearance they can accurately model. PCA-based face analysis typically assumes images of face shape and texture fall in a linear subspace; RBF approaches fare poorly when appearance is not a smooth function.

We want to extend non-rigid interpolation networks to handle cases where appearance is not a linear manifold and is not a smooth function, such as with articulated bodies. The mapping from parameter to appearance for articulated bodies is often one-to-many due to the multiple solutions possible for a given endpoint. It will also be discontinuous when constraints call for different solutions across a boundary in parameter space, such as the example shown in Figure 1.

Our approach represents an appearance mapping as a set of piecewise smooth functions. We search for sets of examples which are well approximated by the examples on the convex hull of the set's parameter values. Once we have these 'safe' sets of examples we perform interpolation using only the examples in a single set.

The clear advantage of this approach is that it will prevent inconsistent examples from being combined during interpolation. It also can reduce the number of examples needed to fully interpolate the function, as only those examples which are on the convex hull of one or more example sets are needed. If a new example is provided and it falls within and is well-approximated by the convex hull of an existing set, it can be safely ignored.

The remainder of this paper proceeds as follows. First, we will review methods for modeling appearance when it can be well approximated with a smooth and/or linear function. Next, we will present a technique for clustering examples to find maximal subsets which are well approximated in their interior. We will then detail how we select among the subsets during interpolation, and finally show results with both synthetic and real imagery.

## 2  Modeling smooth and/or linear appearance functions

Traditional interpolation networks work well when object appearance can be modeled either as a linear manifold or as a smooth function over the parameters of interest (describing pose, expression, identity, configuration, etc.). As mentioned above, both PCA and RBF approaches have been successfully applied to model facial expression.

In both approaches, a key step in modeling non-rigid shape appearance from examples is to couple shape and texture into a single representation. Interpolation of shape has been well studied in the computer graphics literature (e.g., splines for key-frame animation) but does not alone render realistic images. PCA or RBF models of images without a shape model can only represent and interpolate within a very limited range of pose or object configuration.

In a coupled representation, texture is modeled in shape-normalized coordinates, and shape is modeled as disparity between examples or displacement from a canonical example to all examples. Image warping is used to generate images for a particular texture and shape. Given a training set $\Omega = \{(y_i, x_i, d_i), 0 \leq i \leq n\}$, where $y_i$ is the image of example $i$, $x_i$ is the associated pose or configuration parameter, and $d_i$ is a dense correspondence map relative to a canonical pose, a set of shape-aligned texture images can be computed such that texture $t_i$ warped with displacement $d_i$ renders example image $y_i$: $y_i = t_i \circ d_i$ [5, 1, 6]. A new image is constructed using a coupled shape model $G$ and texture model $F$, based on input $u$:

$$\hat{y}(\Omega, u) = F_T(G_D(u), u) ,$$

where $D, T$ are the matrices $[d_0 d_1 ... d_n]$, $[t_0 t_1 ... t_n]$, respectively.

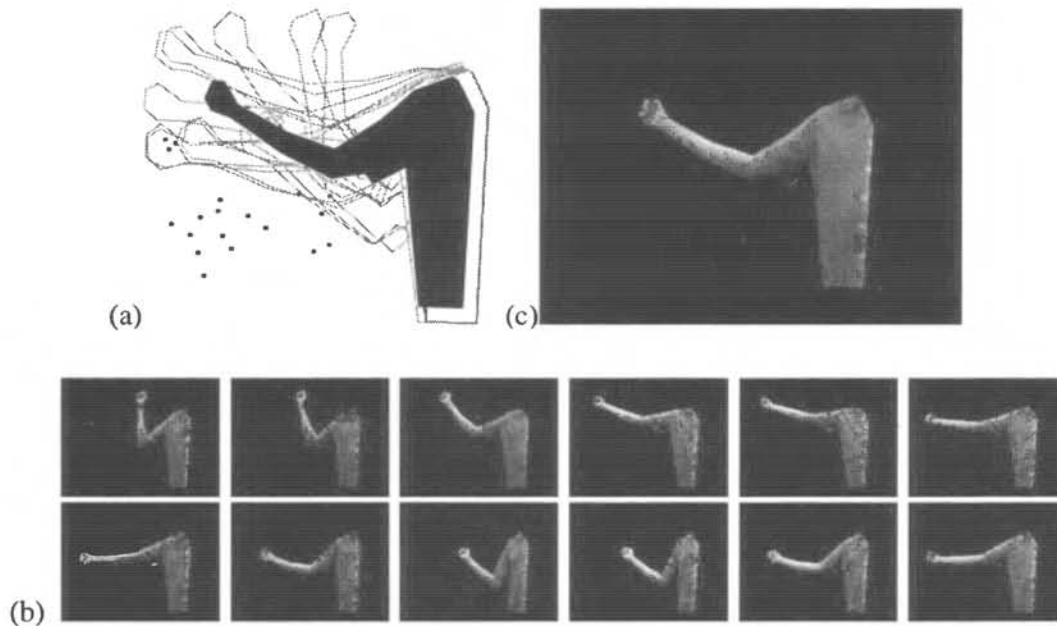

Figure 3: Interpolated shape and texture result. (a) shows exemplar contours (open) and interpolated shape (filled). (b) shows example texture images. (c) shows final interpolated image.

we can use a simple additional constraint on subsequent frames. Once we have selected an example set, we keep using it until the desired parameter value leaves the valid region (convex hull) of that set. When this occurs, we allow transitions only to "adjacent" example sets; adjacency is defined as those pairs of sets for which at least one example on each convex hull are sufficiently close ($||y_i - y_j|| < \epsilon$) in appearance space.

## 5  Results

First we show examples using a synthetic arm with several workspace constraints. Figure 1(a) shows examples of a simple planar 2DOF arm and the inverse kinematic solution for a variety of endpoints. Due to an artificial obstacle in the world, the arm is forced to switch between arm-up and arm-down configurations to avoid collision.

We trained an interpolation network using a single RBF to model the appearance of the arm as a function of endpoint location. Appearance was modeled as the vector of contour point locations, obtained from the synthetic arm rendering function. We first trained a single RBF network on a dense set of examples of this appearance function. Figure 1(b) shows results interpolating new arm images from these examples; results are accurate except where there are regions of appearance discontinuity due to workspace constraints, or when the network extrapolates erroneously.

We applied our clustering method described above to this data, yielding the results shown in Figure 1(c). None of the problems with discontinuities or erroneous extrapolation can be seen in these results, since our method enforces the constraint that an interpolated result must be returned from on or within the convex hull of a valid example set.

Next we applied our method to the images of real arms shown in Figure 2(a). Arm contours were obtained in a sequence of 33 such images using a semi-automated deformable contour tracker augmented with a local image distance metric [3]. Dense correspondences were interpolated from the values on the contour. Figure 2(b) shows interpolated arm shapes using a single RBF on all examples; dramatic errors can be seen near where multiple different

appearances exist within a small region of parameter space.

Figure 2(c) shows the results on the same points using sets of examples found using our clustering method; physically realistic arms are generated in each case. Figure 3 shows the final interpolated result rendered with both shape and texture.

## 6  Conclusion

View-based image interpolation is a powerful paradigm for generating realistic imagery without full models of the underlying scene geometry. Current techniques for non-rigid interpolation assume appearance is a smooth function. We apply an example clustering approach using on-line cross validation to decompose a complex appearance mapping into sets of examples which can be smoothly interpolated. We show results on real imagery of human arms, with correspondences recovered from deformable contour tracking. Given images of an arm moving on a plane with various configuration conditions (elbow up and elbow down), and with associated parameter vectors marking the hand location, our method is able to discover a small set of manifolds with a small number of exemplars each can render new examples which are always physically correct. A single interpolating manifold for this same data has errors near the boundary between different arm configurations, and where multiple images have the same parameter value.

## References

[1]  D. Beymer, A. Shashua and T. Poggio, Example Based Image Analysis and Synthesis, MIT AI Lab Memo No. 1431, MIT, 1993. also see D. Beymer and T. Poggio, *Science* 272:1905-1909, 1996.

[2]  C. Bregler and S. Omohundro, Nonlinear Image Interpolation using Manifold Learning, NIPS-7, MIT Press, 1995.

[3]  T. Darrell, A Radial Cumulative Similarity Transform for Robust Image Correspondence, Proc. CVPR-98, Santa Barbara, CA, IEEE CS Press, 1998.

[4]  M. Jagersand, Image Based View Synthesis of Articulated Agents, Proc. CVPR-97, San Jaun, Pureto Rico, pp. 1047-1053, IEEE CS Press, 1997.

[5]  M. Jones and T. Poggio, Multidimensional Morphable Models, Proc. ICCV-98, Bombay, India, pp. 683-688, 1998.

[6]  A. Lanitis, C.J. Taylor, T.F. Cootes, A Unified Approach to Coding and Interpreting Face Images, Proc. ICCV-95, pp. 368-373, Cambridge, MA, 1995.

[7]  M. Levoy and P. Hanrahan, Light Field Rendering, In SIGGRAPH-96, pp. 31-42, 1996.

[8]  L. McMillan and G. Bishop, Plenoptic Modeling: An image-based rendering system. In Proc. SIGGRAPH-95, pp. 39-46, 1995.

[9]  T. Poggio and F. Girosi, A Theory of Networks for Approximation and Learning, MIT AI Lab Memo No. 1140. 1989.

[10]  T. Rikert and M. Jones, Gaze Estimation using Morphable Models, Proc. IEEE Conf. Face and Gesture Recognition '98, pp. 436-441, Nara, Japan, IEEE CS Press, 1998.

[11]  L. Saul and M. Jordan, A Variational Principle for Model-based Morphing, NIPS-9, MIT Press, 1997.

[12]  S. Seitz and C. Dyer, View Morphing, in Proc. SIGGRAPH-96, pp. 21-30, 1996.

[13]  A. Shashua and M. Werman, Trilinearity of Three Perspective Views and its Associated Tensor, in Proc. ICCV-95, pp. 920-935, Cambridge, MA, IEEE CS Press, 1995.

[14]  J. Tenenbaum, Mapping a manifold of perceptual observations, NIPS-10, MIT Press, 1998.
